# Triangulation by Continuous Embedding

**Marina Meilă and Michael I. Jordan**
{mmp, jordan}@ai.mit.edu
Center for Biological & Computational Learning
Massachusetts Institute of Technology
45 Carleton St. E25-201
Cambridge, MA 02142

## Abstract

When triangulating a belief network we aim to obtain a junction tree of minimum state space. According to (Rose, 1970), searching for the optimal triangulation can be cast as a search over all the permutations of the graph's vertices. Our approach is to embed the discrete set of permutations in a convex continuous domain $D$. By suitably extending the cost function over $D$ and solving the continous nonlinear optimization task we hope to obtain a good triangulation with respect to the aforementioned cost. This paper presents two ways of embedding the triangulation problem into continuous domain and shows that they perform well compared to the best known heuristic.

## 1 INTRODUCTION. WHAT IS TRIANGULATION ?

Belief networks are graphical representations of probability distributions over a set of variables. In what follows it will be always assumed that the variables take values in a finite set and that they correspond to the vertices of a graph. The graph's arcs will represent the dependencies among variables. There are two kinds of representations that have gained wide use: one is the directed acyclic graph model, also called a *Bayes net*, which represents the joint distribution as a product of the probabilities of each vertex conditioned on the values of its parents; the other is the undirected graph model, also called a *Markov field*, where the joint distribution is factorized over the *cliques*[1] of an undirected graph. This factorization is called a *junction tree* and optimizing it is the subject of the present paper. The power of both models lies in their ability to display and exploit existent marginal and conditional independencies among subsets of variables. Emphasizing independencies is useful

from both a qualitative point of view (it reveals something about the domain under study) and a quantitative one (it makes computations tractable). The two models differ in the kinds of independencies they are able to represent and often times in their naturalness in particular tasks. Directed graphs are more convenient for learning a model from data; on the other hand, the clique structure of undirected graphs organizes the information in a way that makes it immediately available to inference algorithms. Therefore it is a standard procedure to construct the model of a domain as a Bayes net and then to convert it to a Markov field for the purpose of querying it.

This process is known as *decomposition* and it consists of the following stages: first, the directed graph is transformed into an undirected graph by an operation called *moralization*. Second, the moralized graph is triangulated. A graph is called *triangulated* if any cycle of length $> 3$ has a *chord* (i.e. an edge connecting two nonconsecutive vertices). If a graph is not triangulated it is always possible to add new edges so that the resulting graph is triangulated. We shall call this procedure *triangulation* and the added edges the *fill-in*. In the final stage, the junction tree (Kjærulff, 1991) is constructed from the maximal cliques of the triangulated graph. We define the state space of a clique to be the cartesian product of the state spaces of the variables associated to the vertices in the clique and we call *weight* of the clique the size of this state space. The *weight of the junction tree* is the sum of the weights of its component cliques. All further exact inference in the net takes place in the junction tree representation. The number of computations required by an inference operation is proportional to the weight of the tree.

For each graph there are several and usually a large number of possible triangulations, with widely varying state space sizes. Moreover, triangulation is the only stage where the cost of inference can be influenced. It is therefore critical that the triangulation procedure produces a graph that is optimal or at least "good" in this respect.

Unfortunately, this is a hard problem. No optimal triangulation algorithm is known to date. However, a number of heuristic algorithms like *maximum cardinality search* (Tarjan and Yannakakis, 1984), *lexicographic search* (Rose et al., 1976) and the *minimum weight heuristic* (MW) (Kjærulff, 1990) are known. An optimization method based on simulated annealing which performs better than the heuristics on large graphs has been proposed in (Kjærulff, 1991) and recently a "divide and conquer" algorithm which bounds the maximum clique size of the triangulated graph has been published (Becker and Geiger, 1996). All but the last algorithm are based on Rose's (Rose, 1970) *elimination procedure*: choose a node $v$ of the graph, connect all its neighbors to form a clique, then eliminate $v$ and all the edges incident to it and proceed recursively. The resulting filled-in graph is triangulated.

It can be proven that the optimal triangulation can always be obtained by applying Rose's elimination procedure with an appropriate ordering of the nodes. It follows then that searching for an optimal triangulation can be cast as a search in the space of all node permutations. The idea of the present work is the following: embed the discrete search space of permutations of $n$ objects (where $n$ is the number of vertices) into a suitably chosen continuous space. Then extend the cost to a smooth function over the continuous domain and thus transform the discrete optimization problem into a continuous nonlinear optimization task. This allows one to take advantage of the thesaurus of optimization methods that exist for continuous cost functions.The rest of the paper will present this procedure in the following sequence: the next section introduces and discusses the objective function; section 3 states the continuous version of the problem; section 4 discusses further aspects of the optimization procedure and presents experimental results and section 5 concludes

the paper.

## 2   THE OBJECTIVE

In this section we introduce the objective function that we used and we discuss its relationship to the junction tree weight. First, some notation. Let $G = (V, E)$ be a graph, its vertex set and its edge set respectively. Denote by $n$ the cardinality of the vertex set, by $r_v$ the number of values of the (discrete) variable associated to vertex $v \in V$, by $\#$ the elimination ordering of the nodes, such that $\#v = i$ means that node $v$ is the $i$-th node to be eliminated according to ordering $\#$, by n$(v)$ the set of neighbors of $v \in V$ in the triangulated graph and by $C_v = \{v\} \cup \{u \in$ n$(v) \mid \#u > \#v\}$.[2] Then, a result in (Golumbic, 1980) allows us to express the total weight of the junction tree obtained with elimination ordering $\#$ as

$$J^*_{(\#)} = \sum_{v \in V} \text{ismax}(C_v) \prod_{u \in C_v} r_u \tag{1}$$

where $\text{ismax}(C_v)$ is a variable which is 1 when $C_v$ is a maximal clique and 0 otherwise. As stated, this is the objective of interest for belief net triangulation. Any reference to optimality henceforth will be made with respect to $J^*$.

This result implies that there are no more than $n$ maximal cliques in a junction tree and provides a method to enumerate them. This suggests defining a cost function that we call the *raw weight* $J$ as the sum over all the cliques $C_v$ (thus possibly including some non-maximal cliques):

$$J_{(\#)} = \sum_{v \in V} \prod_{u \in C_v} r_u \tag{2}$$

$J$ is the cost function that will be used throughout this paper. A reason to use it instead of $J^*$ in our algorithm is that the former is easier to compute and to approximate. How to do this will be the object of the next section. But it is natural to ask first how well do the two agree?

Obviously, $J$ is an upper bound for $J^*$. Moreover, it can be proved that if $r = \min r_v$

$$J^*_{(\#)} \leq J_{(\#)} \leq J^*_{(\#)} \frac{r}{r-1}(1 - \frac{1}{r^n}) \tag{3}$$

and therefore that $J$ is less than a fraction $1/(r-1)$ away from $J^*$. The upper bound is attained when the triangulated graph is fully connected and all $r_v$ are equal.

In other words, the differece between $J$ and $J^*$ is largest for the highest cost triangulation. We also expect this difference to be low for the low cost triangulation. An intuitive argument for this is that good triangulations are associated with a large number of smaller cliques rather than with a few large ones. But the former situation means that there will be only a small number of small size non-maximal cliques to contribute to the difference $J - J^*$, and therefore that the agreement with $J^*$ is usually closer than (3) implies. This conclusion is supported by simulations (Meilă and Jordan, 1997).

## 3   THE CONTINUOUS OPTIMIZATION PROBLEM

This section shows two ways of defining $J$ over continuous domains. Both rely on a formulation of $J$ that eliminates explicit reference to the cliques $C_v$; we describe this formulation here.

Let us first define new variables $\mu_{uv}$ and $e_{uv}$, $u, v = 1, .., n$. For any permutation #

$$\mu_{uv} = \begin{cases} 1 & \text{if } \#u \leq \#v \\ 0 & \text{otherwise} \end{cases} \qquad e_{uv} = \begin{cases} 1 & \text{if the edge } (u, v) \in E \cup F_\# \\ 0 & \text{otherwise} \end{cases}$$

where $F_\#$ is the set of fill-in edges.

In other words, $\mu$ represent precedence relationships and $e$ represent the edges between the $n$ vertices. Therefore, they will be called *precedence* variables and *edge* variables respectively. With these variables, $J$ can be expressed as

$$J_{(\#)} = \sum_{v \in V} \prod_{u \in V} r_u^{\mu_{vu} e_{vu}} \tag{4}$$

In (4), the product $\mu_{vu} e_{vu}$ acts as an indicator variable being 1 iff "$u \in C_v$" is true. For any given permutation, finding the $\mu$ variables is straightforward. Computing the edge variables is possible thanks to a result in (Rose et al., 1976). It states that an edge $(u, v)$ is contained in $F_\#$ iff there is a path in $G$ between $u$ and $v$ containing only nodes $w$ for which $\#w < \min(\#u, \#v)$. Formally, $e_{uv} = e_{vu} = 1$ iff there exists a path $P = (u, w_1, w_2, \ldots v)$ such that

$$\prod_{w_i \in P} \mu_{w_i u} \mu_{w_i v} = 1$$

So far, we have succeeded in defining the cost $J$ associated with any permutation in terms of the variables $\mu$ and $e$. In the following, the set of permutations will be embedded in a continuous domain. As a consequence, $\mu$ and $e$ will take values in the interval $[0, 1]$ but the form of $J$ in (4) will stay the same.

The first method, called $\mu$-continuous embedding ($\mu$-CE) assumes that the variables $\mu_{uv} \in [0, 1]$ represent independent probabilities that $\#u < \#v$. For any permutation, the precedence variables have to satisfy the transitivity condition. Transitivity means that if $\#u < \#v$ and $\#v < \#w$, then $\#u < \#w$, or, that for any triple $(\mu_{uv}, \mu_{vw}, \mu_{wu})$ the assignments $(0, 0, 0)$ and $(1, 1, 1)$ are forbidden. According to the probabilistic interpretation of $\mu$ we introduce a term that penalizes the probability of a transitivity violation:

$$R(\mu) = \sum_{u<v<w} P[(u, v, w) \text{ nontransitive}] \tag{5}$$

$$= \sum_{u<v<w} [\mu_{uv}\mu_{vw}\mu_{wu} + (1 - \mu_{uv})(1 - \mu_{vw})(1 - \mu_{wu})] \tag{6}$$

$$\geq P[\text{assignment non transitive}] \tag{7}$$

In the second approach, called $\theta$-continuous embedding ($\theta$-CE), the permutations are directly embedded into the set of doubly stochastic matrices. A *doubly stochastic matrix* $\theta$ is a matrix for which the elements in a row or column sum to one.

$$\sum_i \theta_{ij} = \sum_j \theta_{ij} = 1 \quad \theta_{ij} \geq 0 \quad \text{for } i, j = 1, ..n. \tag{8}$$

When $\theta_{ij}$ are either 0 or 1, implying that there is exactly one nonzero element in each row or column, the matrix is called a *permutation matrix*. $\theta_{ij} = 1$ and $\#i = j$ both mean that the position of object $i$ is $j$ in the given permutation. The set of doubly stochastic matrices $\Theta$ is a convex polytope of dimension $(n-1)^2$ whose extreme points are the permutation matrices (Balinski and Russakoff, 1974). Thus, every doubly stochastic matrix can be represented as a convex combination of permutation matrices. To constrain the optimum to be a an extreme point, we use the penalty term

$$R(\theta) = \sum_{ij} \theta_{ij}(1 - \theta_{ij}) \tag{9}$$

The precedence variables are defined over $\Theta$ as

$$\mu_{uv} = 1 - \mu_{vu} = \frac{1}{1 - \sum_i \theta_{ui}\theta_{vi}} \sum_{j<i} \theta_{uj}\theta_{vi} \quad u,v,i,j = 1,..n \quad \text{and} \quad v \neq u$$
$$\mu_{vv} = 1$$

Now, for both embeddings, the edge variables can be computed from $\mu$ as follows

$$e_{uv} = e_{vu} = \begin{cases} 1 & \text{for } (u,v) \in E \text{ or } u = v \\ \max_{P \in \{paths\ u \to v\}} \prod_{w \in P} \mu_{wu}\mu_{wv} & \text{otherwise} \end{cases}$$

The above assignments give the correct values for $\mu$ and $e$ for any point representing a permutation. Over the interior of the domain, $e$ is a continuous, piecewise differentiable function. Each $e_{uv}$, $(u,v) \notin E$ can be computed by a shortest path algorithm between $u$ and $v$, with the length of $(w_1, w_2) \in E$ defined as $(-\log \mu_{w_1 u} \mu_{w_2 v})$.

$\theta$-CE is an interior point method whereas in $\mu$-CE the current point, although inside $[0,1]^{n(n-1)/2}$, isn't necessarily in the convex hull of the hypercube's corners that represent permutations. The number of operation required for one evaluation of $J$ and its gradient is as follows: $\mathcal{O}(n^4)$ operations to compute $\mu$ from $\theta$, $\mathcal{O}(n^3 \log n)$ to compute $e$, $\mathcal{O}(n^3)$ for $\frac{\partial J}{\partial e}$ and $\mathcal{O}(n^2)$ for $\frac{\partial J}{\partial \mu}$ and $\frac{\partial J}{\partial \theta}$ afterwards. Since computing $\mu$ is the most computationally intensive step, $\mu$-CE is a clear win in terms of computation cost. In addition, by operating directly in the $\mu$ domain, one level of approximation is eliminated, which makes one expect $\mu$-CE to perform better than $\theta$-CE. The results in the following section will confirm this.

## 4 EXPERIMENTAL RESULTS

To assess the performance of our algorithms we compared their results with the results of the minimum weight heuristic (MW), the heuristic that scored best in empirical tests (Kjærulff, 1990). The lowest junction tree weight obtained in 200 runs of MW was retained and denoted by $J_{MW}^*$. Tests were run on 6 graphs of different sizes and densities:

| graph | h9 | h12 | d10 | m20 | a20 | d20 |
|---|---|---|---|---|---|---|
| $n = |V|$ | 9 | 12 | 10 | 20 | 20 | 20 |
| density | .33 | .25 | .6 | .25 | .45 | .6 |
| $r_{\min}/r_{\max}/r_{\text{avg}}$ | 2/2/2/ | 3/3/3 | 6/15/10 | 2/8/5 | 6/15/10 | 6/15/10 |
| $\log_{10} J_{MW}^*$ | 2.43 | 2.71 | 7.44 | 5.47 | 12.75 | 13.94 |

The last row of the table shows the $\log_{10} J_{MW}^*$. We ran 11 or more trials of each of our two algorithms on each graph. To enforce the variables to converge to a permutation, we minimized the objective $J + \lambda R$, where $\lambda > 0$ is a parameter

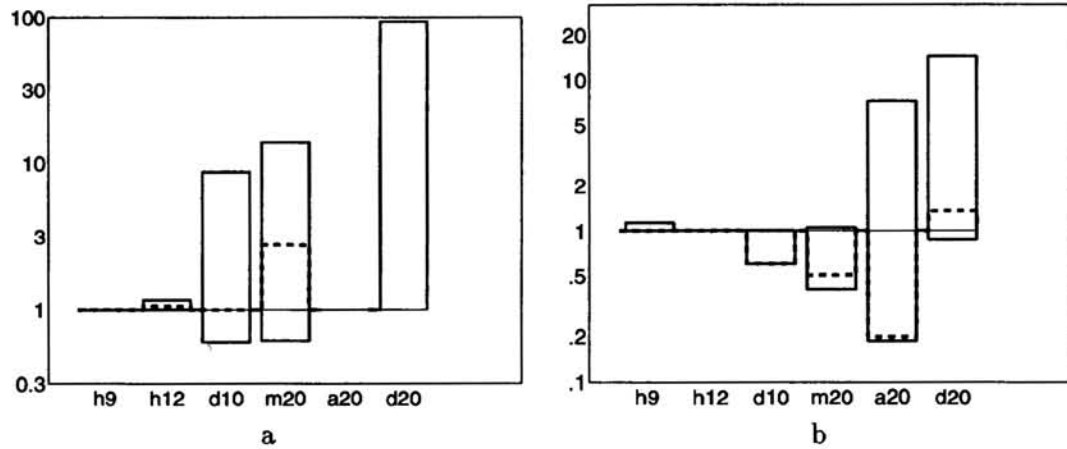

Figure 1: Minimum, maximum (solid line) and median (dashed line) values of $\frac{J^*}{J^*_{MW}}$ obtained by $\theta$-CE (a) and $\mu$-CE (b).

that was progressively increased following a deterministic annealing schedule and $R$ is one of the aforementioned penalty terms. The algorithms were run for 50-150 optimization cycles, usually enough to reach convergence. However, for the $\mu$-embedding on graph **d20**, there were several cases where many $\mu$ values did not converge to 0 or 1. In those cases we picked the most plausible permutation to be the answer.

The results are shown in figure 1 in terms of the ratio of the true cost obtained by the continuous embedding algorithm (denoted by $J^*$) and $J^*_{MW}$. For the first two graphs, **h9** and **h12**, $J^*_{MW}$ is the optimal cost; the embedding algorithms reach it most trials. On the remaining graphs, $\mu$-CE clearly outperforms $\theta$-CE, which also performs poorer than MW on average. On **d10**, **a20** and **m20** it also outperforms the MW heuristic, attaining junction tree weights that are 1.6 to 5 times lower on average than those obtained by MW. On **d20**, a denser graph, the results are similar for MW and $\mu$-CE in half of the cases and worse for $\mu$-CE otherwise. The plots also show that the variability of the results is much larger for CE than for MW. This behaviour is not surprising, given that the search space for CE, although continuous, comprises a large number of local minima. This induces dependence on the initial point and, as a consequence, nondeterministic behaviour of the algorithm. Moreover, while the number of choices that MW has is much lower than the upper limit of $n!$, the "choices" that CE algorithms consider, although soft, span the space of all possible permutations.

# 5   CONCLUSION

The idea of continuous embedding is not new in the field of applied mathematics. The large body of literature dealing with smooth (sygmoidal) functions instead of hard nonlinearities (step functions) is only one example. The present paper shows a nontrivial way of applying a similar treatment to a new problem in a new field. The results obtained by $\mu$-embedding are on average better than the standard MW heuristic. Although not directly comparable, the best results reported on triangulation (Kjærulff, 1991; Becker and Geiger, 1996) are only by little better than ours. Therefore the significance of the latter goes beyond the scope of the present problem. They are obtained on a hard problem, whose cost function has no feature to ease its minimization ($J$ is neither linear, nor quadratic, nor is it additive

w.r.t. the vertices or the edges) and therefore they demonstrate the potential of continuous embedding as a general tool.

Colaterally, we have introduced the cost function $J$, which is directly amenable to continuous approximations and is in good agreement with the true cost $J^*$. Since minimizing $J$ may not be NP-hard, this opens a way for investigating new triangulation methods.

## Acknowledgements

The authors are grateful to Tommi Jaakkola for many discussions and to Ellie Bonsaint for her invaluable help in typing the paper.

## Footnotes

[1] A *clique* is a fully connected set of vertices and a maximal clique is a clique that is not contained in any other clique.

[2]Both n$(v)$ and $C_v$ depend on $\#$ but we chose not to emphasize this in the notation for the sake of readability.

## References

Balinski, M. and Russakoff, R. (1974). On the assignment polytope. *SIAM Rev.*

Becker, A. and Geiger, D. (1996). A sufficiently fast algorithm for finding close to optimal junction trees. In *UAI 96 Proceedings.*

Golumbic, M. (1980). *Algorithmic Graph Theory and Perfect Graphs.* Academic Press, New York.

Kjærulff, U. (1990). Triangulation of graphs–algorithms giving small total state space. Technical Report R 90-09, Department of Mathematics and Computer Science, Aalborg University, Denmark.

Kjærulff, U. (1991). Optimal decomposition of probabilistic networks by simulated annealing. *Statistics and Computing.*

Meilă, M. and Jordan, M. I. (1997). An objective function for belief net triangulation. In Madigan, D., editor, *AI and Statistics*, number 7. (to appear).

Rose, D. J. (1970). Triangulated graphs and the elimination process. *Journal of Mathematical Analysis and Applications.*

Rose, D. J., Tarjan, R. E., and Lueker, E. (1976). Algorithmic aspects of vertex elimination on graphs. *SIAM J. Comput.*

Tarjan, R. and Yannakakis, M. (1984). Simple linear-time algorithms to test chordality of graphs, test acyclicity of hypergraphs, and select reduced acyclic hypergraphs. *SIAM J. Comput.*